# Interior Point Implementations of Alternating Minimization Training

**Michael Lemmon**
Dept. of Electrical Engineering
University of Notre Dame
Notre Dame, IN 46556
lemmon@maddog.ee.nd.edu

**Peter T. Szymanski**
Dept. of Electrical Engineering
University of Notre Dame
Notre Dame, IN 46556
pszymans@maddog.ee.nd.edu

## Abstract

This paper presents an alternating minimization (AM) algorithm used in the training of radial basis function and linear regressor networks. The algorithm is a modification of a small-step interior point method used in solving primal linear programs. The algorithm has a convergence rate of $O(\sqrt{n}L)$ iterations where $n$ is a measure of the network size and $L$ is a measure of the resulting solution's accuracy. Two results are presented that specify how aggressively the two steps of the AM may be pursued to ensure convergence of each step of the alternating minimization.

## 1  Introduction

In recent years, considerable research has investigated the use of alternating minimization (AM) techniques in the supervised training of radial basis function networks. AM techniques were first introduced in soft-competitive learning algorithms[1]. This training procedure was later shown to be closely related to Expectation-Maximization algorithms used by the statistical estimation community[2]. Alternating minimizations search for optimal network weights by breaking the search into two distinct minimization problems. A given network performance functional is extremalized first with respect to one set of network weights and then with respect to the remaining weights. These learning procedures have found applications in the training of local expert systems [3], and in Boltzmann machine training [4]. More recently, convergence rates have been derived by viewing the AM

method as a variable metric algorithm [5].

This paper examines AM as a perturbed linear programming (LP) problem. Recent advances in the application of barrier function methods to LP problems have resulted in the development of "path following" or "interior point" (IP) algorithms [6]. These algorithms are characterized by a fast convergence rate that scales in a sublinear manner with problem size. This paper shows how a *small-step* IP algorithm for solving a primal LP problem can be modified into an AM training procedure.. The principal results of the paper are bounds on how aggressively the legs of the alternating minimization can be pursued so that the AM algorithm maintains the sublinear convergence rate characteristic of its LP counterpart and so that both legs converge to an optimal solution.

## 2   Problem Statement

Consider a function approximation problem where a stochastic approximator learns a mapping $f : \mathbb{R}^N \to \mathbb{R}$. The approximator computes a *predicted output*, $\hat{y} \in \mathbb{R}$, given the input $\mathbf{z} \in \mathbb{R}^N$. The prediction is computed using a finite set of $M$ regressors. The $m^{th}$ regressor is characterized by the pair $(\theta_m, \omega_m) \in \mathbb{R}^N \times \mathbb{R}^N$ $(m = 1, \ldots, M)$. The output of the $m^{th}$ regressor, $\hat{y}_m \in \mathbb{R}$, in response to an input, $\mathbf{z} \in \mathbb{R}^N$ is given by the linear function

$$\hat{y}_m = \theta_m^T \mathbf{z}. \tag{1}$$

The $m^{th}$ regressor $(m = 1, \ldots, M)$ has an associated radial basis function (RBF) with parameter vector $\omega_m \in \mathbb{R}^N$. The $m^{th}$ RBF weights the contribution of the $m^{th}$ output in computing $\hat{y}$ and is defined as a normal probability density function

$$Q(m|\mathbf{z}) = k_m \exp(-\sigma^{-2}||\omega_m - \mathbf{z}||^2) \tag{2}$$

where $k_m$ is a normalizing constant. The set of all weights or *gating probabilities* is denoted by $Q$. The parameterization of the $m^{th}$ regressor is $\Theta_m = \langle \theta_m^T, \omega_m^T \rangle^T \in \mathbb{R}^{2N}$ $(m = 1, \ldots, M)$ and the parameterization of the set of $M$ linear regressors is

$$\Theta = \langle \Theta_1^T, \ldots, \Theta_M^T \rangle^T. \tag{3}$$

The preceding stochastic approximator can be viewed as a neural network. The network consists of $M + 1$ neurons. $M$ of the neurons are *agent neurons* while the other neuron is referred to as a *gating neuron*. The $m^{th}$ agent neuron is parameterized by $\theta_m$, the first element of the pair $\Theta_m = \langle \theta_m^T, \omega_m^T \rangle^T$ $(m = 1, \ldots, M)$. The agent neurons receive as input, the vector $\mathbf{z} \in \mathbb{R}^N$. The output of the $m^{th}$ agent neuron in response to an input $\mathbf{z}$ is $\hat{y}_m = \theta_m^T \mathbf{z}$ $(m = 1, \ldots, M)$. The gating neuron is parameterized by the conditional gating probabilities, $Q$. The gating probabilities are defined by the set of vectors, $\bar{\omega} = \{\omega_1, \ldots, \omega_M\}$. The gating neuron receives the agent neurons' outputs and the vector $\mathbf{z}$ as its input. The gating neuron computes the network's output, $\hat{y}$, as a hard (4) or soft (5) choice

$$\hat{y} = \hat{y}_m; \; m = \arg \max_{m=1,\ldots,M} Q(m|\mathbf{z}) \tag{4}$$

$$\hat{y} = \frac{\sum_{m=1}^M Q(m|\mathbf{z}) \hat{y}_m}{\sum_{m=1}^M Q(m|\mathbf{z})}. \tag{5}$$

The network will be said to be "optimal" with respect to a training set $\mathcal{T} = \{(\mathbf{z}_i, y_i) : y_i = f(\mathbf{z}_i), i = 1, \ldots, R\}$ if a mean square error criterion is minimized. Define the square output error of the $m^{th}$ agent neuron to be $e_m(\mathbf{z}_i) = (y_i - \theta_m^T \mathbf{z}_i)^2$ and the square weighting or *classifier* error of the $m^{th}$ RBF to be $c_m(\mathbf{z}_i) = \|\omega_m - \mathbf{z}_i\|^2$. Let the combined square approximation error of the $m^{th}$ neuron be $d_m(\mathbf{z}_i) = \kappa_e e_m(\mathbf{z}_i) + \kappa_c c_m(\mathbf{z}_i)$ and let the average square approximation error of the network be

$$\bar{d}(Q, \Theta, \mathcal{T}) = \sum_{m=1}^{M} \sum_{i=1}^{R} p(\mathbf{z}_i) Q(m|\mathbf{z}_i) d_m(\mathbf{z}_i). \tag{6}$$

Minimizing (6) corresponds to minimizing both the output error in the $M$ linear regressors and the classifier error in assigning inputs to the $M$ regressors. Since $\mathcal{T}$ is a discrete set and the $Q$ are gating probabilities, the minimization of $\bar{d}(Q, \Theta, \mathcal{T})$ is constrained so that $Q$ is a valid conditional probability mass function over the training set, $\mathcal{T}$.

Network training can therefore be viewed as a constrained optimization problem. In particular, this optimization problem can be expressed in a form very similar to conventional LP problems. The following notational conventions are adopted to highlight this connection. Let $\mathbf{x} \in \mathbb{R}^{MR}$ be the gating neuron's weight vector where

$$\mathbf{x} = \langle Q(1|\mathbf{z}_1), \ldots, Q(1|\mathbf{z}_R), Q(2|\mathbf{z}_1), \ldots, Q(m|\mathbf{z}_i), \ldots \rangle^T. \tag{7}$$

Let $\Theta_m = \langle \theta_m^T, \omega_m^T \rangle^T \in \mathbb{R}^{2N}$ denote the parameter vectors associated with the $m^{th}$ regressor and define the cost vector conditioned on $\Theta = \langle \Theta_1^T, \ldots, \Theta_M^T \rangle^T$ as

$$\mathbf{c}(\Theta) = \langle p(\mathbf{z}_1) d_1(\mathbf{z}_1), \ldots, p(\mathbf{z}_R) d_1(\mathbf{z}_R), p(\mathbf{z}_2) d_2(\mathbf{z}_2), \ldots, p(\mathbf{z}_i) d_m(\mathbf{z}_i), \ldots \rangle^T \tag{8}$$

With this notation, the network training problem can be stated as follows,

$$\begin{array}{ll} \text{minimize} & \mathbf{c}^T(\Theta)\mathbf{x} \\ \text{with respect to} & \mathbf{x}, \Theta \\ \text{subject to} & A\mathbf{x} = \mathbf{b}, \mathbf{x} \geq 0 \end{array} \tag{9}$$

where $\mathbf{b} = (1, \cdots, 1)^T \in \mathbb{R}^R$, $A = [I_{R \times R} \cdots I_{R \times R}] \in \mathbb{R}^{R \times MR}$, and $\mathbf{x} \geq 0$ implies $x_i \geq 0$ for $i = 1, \ldots, MR$.

One approach for solving this problem is to break up the optimization into two steps. The first step involves minimizing the above cost functional with respect to $\mathbf{x}$ assuming a fixed $\Theta$. This is the $Q$-update of the algorithm. The second leg of the algorithm minimizes the functional with respect to $\Theta$ assuming fixed $\mathbf{x}$. This leg is called the $\Theta$-update. Because the proposed optimization alternates between two different subsets of weights, this training procedure is often referred to as *alternating minimization*. Note that the $Q$-update is an LP problem while the $\Theta$-update is a quadratic programming problem. Consequently, the AM training procedure can be viewed as a perturbed LP problem.

## 3   Proposed Training Algorithm

The preceding section noted that network training can be viewed as a perturbed LP problem. This observation is significant for there exist very efficient LP solvers

based on barrier function methods used in non-linear optimization. Recent advances in path following or interior point (IP) methods have developed LP solvers which exhibit convergence rates which scale in a sublinear way with problem size [6]. This section introduces a modification of a small-step primal IP algorithm that can be used for neural network training. The proposed modification is later shown to preserve the computational efficiency enjoyed by its LP counterpart.

To see how such a modification might arise, we first need to examine path following LP solvers. Consider the following LP problem.

$$
\begin{array}{ll}
\text{minimize} & \mathbf{c}^T \mathbf{x} \\
\text{with respect to} & \mathbf{x} \in \mathbb{R}^n \\
\text{subject to} & A\mathbf{x} = \mathbf{b}, \mathbf{x} \geq 0
\end{array} \tag{10}
$$

This problem can be solved by solving a sequence of augmented optimization problems arising from the primal parameterization of the LP problem.

$$
\begin{array}{ll}
\text{minimize} & \alpha^{(k)} \mathbf{c}^T \mathbf{x}^{(k)} - \sum_i \log x_i^{(k)} \\
\text{with respect to} & \mathbf{x}^{(k)} \in \mathbb{R}^n \\
\text{subject to} & A\mathbf{x}^{(k)} = \mathbf{b}, \mathbf{x}^{(k)} \geq 0
\end{array} \tag{11}
$$

where $\alpha^{(k)} \geq 0$ $(k = 1, \cdots, K)$ is a finite length, monotone increasing sequence of real numbers. $\mathbf{x}^*(\alpha^{(k)})$ denotes the solution for the $k$th optimization problem in the sequence and is referred to as a *central point*. The locus of all points, $\mathbf{x}^*(\alpha^{(k)})$ where $\alpha^{(k)} \geq 0$ is called the *central path*. The augmented problem takes the original LP cost function and adds a logarithmic barrier which keeps the central point away from the boundaries of the feasible set. As $\alpha$ increases, the effect of the barrier is decreased, thereby allowing the $k^{th}$ central point to approach the LP problem's solution in a controlled manner.

Path following (IP) methods solve the LP problem by approximately solving the sequence of augmented problems shown in (11). The parameter sequence, $\alpha^{(0)}, \alpha^{(1)}, \cdots, \alpha^{(K)}$, is chosen to be a monotone increasing sequence so that the central points, $\mathbf{x}^*(\alpha^{(k)})$, of each augmented optimization approach the LP solution in a monotone manner. It has been shown that for specific choices of the $\alpha$ sequence, that the sequence of approximate central points will converge to an $\epsilon$-neighborhood of the LP solution after a finite number of iterations. For primal IP algorithms, the required condition is that successive approximations of the central points lie within the region of quadratic convergence for a scaling steepest descent (SSD) algorithm [6]. In particular, it has been shown that if the $k^{th}$ approximating solution is sufficiently close to the $k^{th}$ central point and if $\alpha^{(k+1)} = \alpha^{(k)}(1 + \nu/\sqrt{n})$ where $\nu \leq 0.1$ controls the distance between successive central points, then the "closeness" to the $(k + 1)^{st}$ central point is guaranteed and the resulting algorithm will converge in $O(\sqrt{n}L)$ iterations where $L = n + p + 1$ specifies the size of the LP problem and $p$ is the total number of bits used to represent the data in $A$, $\mathbf{b}$, and $\mathbf{c}$. If the algorithm takes small steps, then it is guaranteed to converge efficiently.

The preceding discussion reveals that a key component to a path following method's computational efficiency lies in controlling the iteration so that successive central points lie within the SSD algorithm's region of quadratic convergence. If we are to successfully extend such methods to (9), then this "closeness" of successive solutions must be preserved by the $\Theta$-update of the algorithm. Due to the quadratic nature

of the $\Theta$-update, this minimization can be done exactly using a single Newton-Raphson iteration. Let $\Theta^*$ denote $\Theta$-update's minimizer. If we update $\Theta$ to $\Theta^*$, it is quite possible that the cost vector, $\mathbf{c}(\Theta)$, will be rotated in such a way that the current solution, $\mathbf{x}^{(k)}$, no longer lies in the region of quadratic convergence. Therefore, if we are to preserve the IP method's computational efficiency it will be necessary to be less "aggressive" in the $\Theta$-update. In particular, this paper proposes the following convex combination as the $\Theta$-update

$$\Theta_m^{(k+1)} = (1 - \gamma^{(k)})\Theta_m^{(k)} + \gamma^{(k)}\Theta_m^{(k+1),*} \tag{12}$$

where $\Theta_m^{(k)}$ is the $m^{th}$ parameter vector at time $k$ and $0 < \gamma^{(k)} < 1$ controls the size of the update. This will ensure convergence of the $Q$-update.

Convergence of the AM algorithm also requires convergence of the $\Theta$-update. For the $\Theta$-update to converge, $\gamma^{(k)}$ in (12) must go to unity as $k$ increases. Convergence of $\gamma^{(k)}$ to unity requires that the sequence $||\Theta_m^{(k+1),*} - \Theta_m^{(k)}||$ be monotone decreasing. As the $\Theta$-update minimizer, $\Theta^{(k+1),*}$, depends upon the current weights, $Q^{(k)}(m|\mathbf{z})$, large changes to $Q$ can prevent the sequence from being monotone decreasing. Thus, it is necessary to also be less "aggressive" in the $Q$-update. An appropriate bound on $\nu$ is the proposed solution to guarantee convergence of the $\Theta$-update.

---

**Algorithm 1 (Proposed Training Algorithm)**

Initialize
    $k = 0$.
    Choose $x_i^{(k)}$, $\Theta_m^{(k)}$, and $\alpha^{(k)}$ for $i = 1, \cdots, (MR)$, and for $m = 1, \cdots, M$.
repeat
    $\alpha^{(k+1)} = \alpha^{(k)}(1 + \nu/\sqrt{n})$, where $\nu \leq 0.1$.
    $Q$-update:
        $\mathbf{x}_0 = \mathbf{x}^{(k)}$
        for $i = 0, \cdots, P - 1$
            $\mathbf{x}_{i+1} = ScalingSteepestDescent(\mathbf{x}_i^{(k+1)}, \alpha^{(k+1)}, \Theta^{(k)})$
        $\mathbf{x}^{(k+1)} = \mathbf{x}_P$
    $\Theta$-update:  For $m = 1, \ldots, M$

$$\Theta_m^{(k+1)} = (1 - \gamma^{(k)})\Theta_m^{(k)} + \gamma^{(k)}\Theta_m^{(k+1),*}$$

    $k = k + 1$
until$(\Delta < \epsilon)$

---

## 4   Theoretical Results

This section provides bounds on the parameter, $\gamma^{(k)}$, controlling the AM algorithm's $\Theta$-update so that successive $\mathbf{x}^{(k)}$ vector solutions lie within the SSD algorithm's region of quadratic convergence and on $\nu$ controlling the $Q$-update so that successive central points are not too far apart, thus allowing convergence of the $\Theta$-update.

**Theorem 1** *Let $\Theta_m^{(k)}$ and $\Theta_m^{(k),*}$ be the current and minimizing parameter vectors at time $k$. Let $\mathbf{c}^{(k)} = \mathbf{c}(\Theta^{(k)})$ and $\mathbf{c}^{(k),*} = \mathbf{c}(\Theta^{(k),*})$. Let $\delta(\mathbf{x}, \alpha, \Theta) =$*

$||P_{AX}X\left(\alpha c(\Theta) - x^{-1}\right)||$ *be the step size of the SSD update where* $P_A = I - A^T(AA^T)^{-1}A$ *and* $X = diag(x_1, \ldots, x_n)$. *Assume that* $\delta(\mathbf{x}^{(k+1)}, \alpha^{(k+1)}, \Theta^{(k)}) = \delta_1 < 0.5$ *and let* $\Theta_m^{(k+1)} = (1 - \gamma^{(k)})\Theta_m^{(k)} + \gamma^{(k)}\Theta_m^{(k+1),*}$. *If* $\gamma^{(k)}$ *is chosen as*

$$\gamma^{(k)} \leq \frac{\delta_2 - \delta_1}{n(1 + \nu/\sqrt{n})||\mathbf{c}^{(k+1),*} - \mathbf{c}^{(k)}||} \tag{13}$$

*where* $\delta_1 < \delta_2 = 0.5$, *then* $\delta(\mathbf{x}^{(k+1)}, \alpha^{(k+1)}, \Theta^{(k+1)}) \leq \delta_2 = 0.5$.

**Proof:** The proof must show that the choice of $\gamma^{(k)}$ maintains the nearness of $\mathbf{x}^{(k+1)}$ to the central path after the $\Theta$-update. Let $\mathbf{h}(\mathbf{x}, \alpha, \Theta) = P_{AX}X\left(\alpha c(\Theta) - \mathbf{x}^{-1}\right)$, $\mathbf{h}_1 = \mathbf{h}(\mathbf{x}^{(k+1)}, \alpha^{(k+1)}, \Theta^{(k)})$ and $\mathbf{h}_2 = \mathbf{h}(\mathbf{x}^{(k+1)}, \alpha^{(k+1)}, \Theta^{(k+1)})$. Using the triangle inequality produces

$$||\mathbf{h}_2|| \leq ||\mathbf{h}_2 - \mathbf{h}_1|| + ||\mathbf{h}_1||.$$

$||\mathbf{h}_1|| = \delta_1$ by assumption, so

$$||\mathbf{h}_2|| \leq ||\alpha^{(k+1)}P_{AX}X(\mathbf{c}^{(k+1)} - \mathbf{c}^{(k)})|| + \delta_1.$$

Using the convexity of the cost vectors produces $(\mathbf{c}^{(k+1)} - \mathbf{c}^{(k)}) \leq (1 - \gamma^{(k)})\mathbf{c}^{(k)} + \gamma^{(k)}\mathbf{c}^{(k+1),*} - \mathbf{c}^{(k)}$ resulting in

$$||\mathbf{h}_2|| \leq ||\alpha^{(k+1)}\gamma^{(k)}P_{AX}X(\mathbf{c}^{(k+1),*} - \mathbf{c}^{(k)})|| + \delta_1.$$

Using the fact that $||P_{AX}X|| \leq \Delta = n/\alpha^{(k)}$ ($\Delta$ is the *duality gap*),

$$\begin{aligned}||\mathbf{h}_2|| &\leq \gamma^{(k)}\alpha^{(k+1)}||P_{AX}X||\ ||\mathbf{c}^{(k+1),*} - \mathbf{c}^{(k)}|| + \delta_1. \\ &\leq \gamma^{(k)}n(1 + \nu/\sqrt{n})||\mathbf{c}^{(k+1),*} - \mathbf{c}^{(k)}|| + \delta_1.\end{aligned}$$

Plugging in the value of $\gamma^{(k)}$ from (13) and simplifying produces the desired result $||\mathbf{h}_2|| \leq \delta_2 \leq 0.5$, guaranteeing that $\mathbf{x}^{(k+1)}$ remains close to $\mathbf{x}^{(k+1),*}$ after the $\Theta$-update.  $\square$

Theorem 1 shows that the non-linear optimization can be embedded within the steps of the path following algorithm without it taking solutions too far from successive central points. The following two results, found in [7], provide a bound on $\nu$ to guarantee convergence of the $\Theta$-update. The bound on $\nu$ forces successive central points to be close and ensures convergence of the $\Theta$-update.

**Proposition 1** *Let* $B = \sum_{\mathbf{z}} p_{\mathbf{z}}Q(m|\mathbf{z})\mathbf{z}\mathbf{z}^T$, $E = \sum_{\mathbf{z}} p_{\mathbf{z}}\Delta Q(m|\mathbf{z})\mathbf{z}\mathbf{z}^T$, $\mathbf{w} = \sum_{\mathbf{z}} p_{\mathbf{z}}Q(m|\mathbf{z})y(\mathbf{z})\mathbf{z}$, *and* $\Delta\mathbf{w} = \sum_{\mathbf{z}} p_{\mathbf{z}}\Delta Q(m|\mathbf{z})y(\mathbf{z})\mathbf{z}$, *where* $Q(m|\mathbf{z}) = Q^{(k)}(m|\mathbf{z})$ *and* $\Delta Q(m|\mathbf{z}) = Q^{(k+1)}(m|\mathbf{z}) - Q^{((k)}m|\mathbf{z})$. *Assume that* $|y(\mathbf{z})| \leq Y$, $||\mathbf{z}|| \leq \zeta$, *and that* $B$ *is of full rank for all valid Q's. Finally, let* $\mu_{max} = \sup_Q ||B^{-1}||$. *Then,*

$$||\Theta_m^{(k+1),*} - \Theta_m^{(k),*}|| \leq 2(\nu^2 + 2\nu)K \tag{14}$$

*where* $K = \mu_{max}Y\zeta\left(1 + \frac{\zeta^2\mu_{max}}{1-r}\right)$ *and* $r = ||B^{-1}||\ ||E|| < 1$.

**Theorem 2** *Assume that* $||\mathbf{z}|| \leq \zeta$, $|y(\mathbf{z})| \leq Y$, $||\Theta_m|| \leq \Theta_{max}$, *and that* $B = \sum_{\mathbf{z}} p_{\mathbf{z}}Q(m|\mathbf{z})\mathbf{z}\mathbf{z}^T$ *is of full rank for all valid Q's and that* $||B^{-1}||\ ||E|| = r < 1$. *If*

$$\begin{aligned}\nu \leq \min\{&0.1, \\ &-1 + \sqrt{1 + r/(2\zeta^2\mu_{max})}, \\ &-1 + \sqrt{1 + \gamma_{min}\epsilon_\Theta/(2K)}\}\end{aligned} \tag{15}$$

*where* $K = \mu_{max} Y \zeta (1 + \zeta^2 \mu_{max}/(1 - r))$, $\gamma_{min} = (\delta_2 - \delta_1)/(n(1 + 0.1/\sqrt{n})\|c^{(1),*} - c^{(0)}\|)$ *and* $\epsilon_\Theta$ *is the largest* $\|\Theta_m^{(k+1),*} - \Theta_m^{(k)}\|$ *such that* $\gamma^{(k)} = 1$, *then the* $\Theta$*-update will converge with* $\|\Theta_m^{(k+1),*} - \Theta_m^{(k)}\| \to 0$ *and* $\gamma^{(k)} \to 1$ *as* $k$ *increases.*

The preceding results guarantee the convergence of the component minimizations separately. Convergence of the total algorithm relies on the simultaneous convergence of both steps. This is currently being addressed using contraction mapping concepts and stability results from nonlinear stability analysis [8].

The convergence rate of the algorithm is established using the LP problem's duality gap. The duality gap is the difference between the current solutions for the primal and dual formulations of the LP problem. Path following algorithms allow the duality gap to be expressed as follows

$$\Delta(\alpha^{(k)}) = \frac{n + 0.5\sqrt{n}}{\alpha^{(k)}}. \tag{16}$$

and thus provide a convenient stopping criterion for the algorithm. Note that $\alpha^{(k)} = \alpha^{(0)}/\beta^k$ where $\beta \leq (1 + \nu/\sqrt{n})$. This implies that $\Delta^{(k)} = \beta^k \Delta^{(0)} \leq \beta^k 2^L$. If $k$ is chosen so that $\beta^k 2^L \leq 2^{-L}$, then $\Delta^{(k)} \leq 2^{-L}$ which implies that $k \geq 2L/\log(1/\beta)$. Inserting our choice of $\bar{\beta}$ one finds that $k \geq (2\sqrt{n}L/\nu) + 2L$. The preceding argument establishes that the proposed convergence rate of $O(\sqrt{n}L)$ iterations. In other words, the procedure's training time scales in a *sublinear* manner with network size.

## 5   Simulation Example

Simulations were run on a time series prediction task to test the proposed algorithm. The training set is $T = \{(z_i, y_i) : y_i = y(iT), z_i = \langle y_{i-1}, y_{i-2}, \ldots, y_{i-N} \rangle^T \in \mathbb{R}^N\}$ for $i = 0, 1, \ldots, 100$, $N = 4$, and $T = 0.04$ where the time series is defined as

$$y(t) = \sin(\pi t) - \sin(2\pi t) + \sin(3\pi t) - \sin(\pi t/2) \tag{17}$$

The results describe the convergence of the algorithm. These experiments consisted of 100 randomly chosen samples with $N = 4$ and a number of agent neurons ranging from $M = 4$ to 20. This corresponds to an LP problem dimension of $n = 404$ to 2020. The stopping criteria for the tests was to run until the solution was within $\epsilon = 10^{-3}$ of a local minimum. The number of iterations and floating point operations (FLOPS) for the AM algorithm to converge are shown in Figures 1(a) and 1(b) with AM results denoted by "o" and the theoretical rates by a solid line. The algorithm exhibits approximately $O(\sqrt{n}L)$ iterations to converge as predicted. The computational cost, however is $O(n^2 L)$ FLOPS which is better than the predicted $O(n^{3.5}L)$. The difference is due to the use of sparse matrix techniques which reduce the number of computations. The resulting AM algorithm then has the complexity of a *matrix multiplication* instead of a matrix inversion. The use of the algorithm resulted in networks having mean square errors on the order of $10^{-3}$.

## 6   Discussion

This paper has presented an AM algorithm which can be proven to converge in $O(\sqrt{n}L)$ iterations. The work has established a means by which IP methods can be

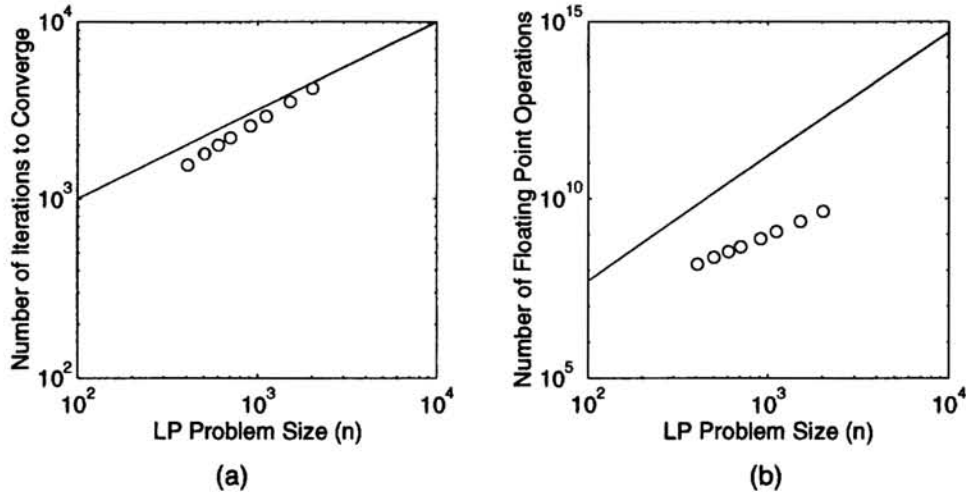

Figure 1: Convergence rates as a function of $n$

applied to NN training in a way which preserves the computational efficiency of IP solvers. The AM algorithm can be used to solve off-line problems such as codebook generation and parameter identification in colony control applications. The method is currently being used to solve hybrid control problems of the type in [9]. Areas of future research concern the study of large-step IP methods and extensions of AM training to other EM algorithms.

# References

[1] S. Nowlan, "Maximum likelihood competitive learning," in *Advances in Neural Information Processing Systems 2*, pp. 574–582, San Mateo, California: Morgan Kaufmann Publishers, Inc., 1990.

[2] M. Jordan and R. Jacobs, "Hierarchical mixtures of experts and the EM algorithm," Tech. Rep. 9301, MIT Computational Cognitive Science, Apr. 1993.

[3] R. Jacobs, M. Jordan, S. Nowlan, and G. Hinton, "Adaptive mixtures of local experts," *Neural Computation*, vol. 3, pp. 79–87, 1991.

[4] W. Byrne, "Alternating minimization and Boltzmann machine learning," *IEEE Transactions on Neural Networks*, vol. 3, pp. 612–620, July 1992.

[5] M. Jordan and L. Xu, "Convergence results for the EM approach to mixtures of experts architectures," Tech. Rep. 9303, MIT Computational Cognitive Science, Sept. 1993.

[6] C. Gonzaga, "Path-following methods for linear programming," *SIAM Review*, vol. 34, pp. 167–224, June 1992.

[7] P. Szymanski and M. Lemmon, "A modified interior point method for supervisory controller design," in *Proceedings of the 33rd IEEE Conference on Decision and Control*, pp. 1381–1386, Dec. 1994.

[8] M. Vidyasagar, *Nonlinear Systems Analysis*. Englewood Cliffs, New Jersey: Prentice-Hall, Inc., 1993.

[9] M. Lemmon, J. Stiver, and P. Antsaklis, "Event identification and intelligent hybrid control," in *Hybrid Systems* (R. L. Grossman, A. Nerode, A. P. Ravn, and H. Rischel, eds.), vol. 736 of *Lecture Notes in Computer Science*, pp. 265–296, Springer-Verlag, 1993.
